# Gradient Flow Independent Component Analysis in Micropower VLSI

**Abdullah Celik, Milutin Stanacevic and Gert Cauwenberghs**
Johns Hopkins University, Baltimore, MD 21218
{acelik,miki,gert}@jhu.edu

## Abstract

We present micropower mixed-signal VLSI hardware for real-time blind separation and localization of acoustic sources. Gradient flow representation of the traveling wave signals acquired over a miniature (1cm diameter) array of four microphones yields linearly mixed instantaneous observations of the time-differentiated sources, separated and localized by independent component analysis (ICA). The gradient flow and ICA processors each measure 3mm × 3mm in 0.5 $\mu$m CMOS, and consume 54 $\mu$W and 180 $\mu$W power, respectively, from a 3 V supply at 16 ks/s sampling rate. Experiments demonstrate perceptually clear (12dB) separation and precise localization of two speech sources presented through speakers positioned at 1.5m from the array on a conference room table. Analysis of the multipath residuals shows that they are spectrally diffuse, and void of the direct path.

## 1  Introduction

Time lags in acoustic wave propagation provide cues to localize an acoustic source from observations across an array. The time lags also complicate the task of separating multiple co-existing sources using independent component analysis (ICA), which conventionally assumes instantaneous mixture observations.

Inspiration from biology suggests that for very small aperture (spacing between acoustic sensors *i.e.,* tympanal membranes), small differences (gradients) in sound pressure *level* are more effective in resolving source direction than actual (microsecond scale) time differences. The remarkable auditory localization capability of certain insects at a small (1%) fraction of the wavelength of the source owes to highly sensitive differential processing of sound pressure through inter-tympanal mechanical coupling [1] or inter-aural coupled neural circuits [2].

We present a mixed-signal VLSI system that operates on spatial and temporal differences (gradients) of the acoustic field at very small aperture to separate and localize mixtures of traveling wave sources. The real-time performance of the system is characterized through experiments with speech sources presented through speakers in a conference room setting.

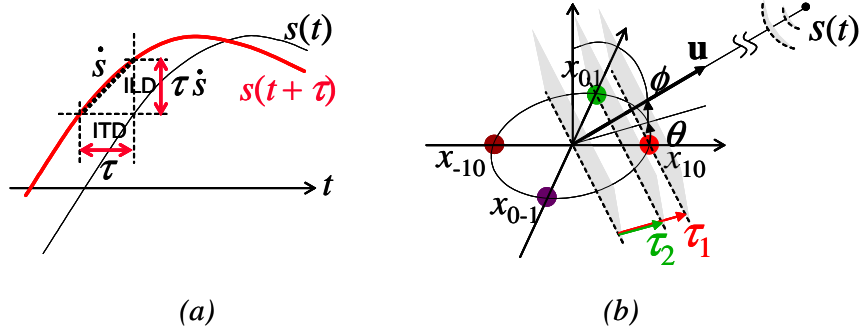

<center>(a)</center> <center>(b)</center>

Figure 1: *(a)* Gradient flow principle. At low aperture, interaural level differences (ILD) and interaural time differences (ITD) are directly related, scaled by the temporal derivative of the signal. *(b)* 3-D localization (azimuth $\theta$ and elevation $\phi$) of an acoustic source using a planar geometry of four microphones.

## 2 Gradient Flow Independent Component Analysis

Gradient flow [3, 4] is a signal conditioning technique for source separation and localization suited for arrays of very small aperture, *i.e.,* of dimensions significantly smaller than the shortest wavelength in the sources. The principle is illustrated in Figure 1 (a). Consider a traveling acoustic wave impinging on an array of four microphones, in the configuration of Figure 1 (b). The 3-D direction cosines of the traveling wave $\mathbf{u}$ are implied by propagation delays $\tau_1$ and $\tau_2$ in the source along directions $p$ and $q$ in the sensor plane. Direct measurement of these delays is problematic as they require sampling in excess of the bandwidth of the signal, increasing noise floor and power requirements. However, indirect estimates of the delays are obtained, to first order, by relating spatial and temporal derivatives of the acoustic field:

$$\begin{aligned} \xi_{10}(t) &\approx \tau_1 \dot{\xi}_{00}(t) \\ \xi_{01}(t) &\approx \tau_2 \dot{\xi}_{00}(t) \end{aligned} \qquad (1)$$

where $\xi_{10}$ and $\xi_{01}$ represent spatial gradients in $p$ and $q$ directions around the origin ($p = q = 0$), $\xi_{00}$ the spatial common mode, and $\dot{\xi}_{00}$ its time derivative. Estimates of $\xi_{00}$, $\xi_{10}$ and $\xi_{01}$ for the sensor geometry of Figure 1 can be obtained as:

$$\begin{aligned} \xi_{00} &\approx \tfrac{1}{4}\big(x_{-1,0} + x_{1,0} + x_{0,-1} + x_{0,1}\big) \\ \xi_{10} &\approx \tfrac{1}{2}\big(x_{1,0} - x_{-1,0}\big) \\ \xi_{01} &\approx \tfrac{1}{2}\big(x_{0,1} - x_{0,-1}\big) \end{aligned} \qquad (2)$$

A single source can be localized by estimating direction cosines $\tau_1$ and $\tau_2$ from (1), a principle known for years in monopulse radar, exploited by parasite insects [1], and implemented in mixed-signal VLSI hardware [6]. As shown in Figure 1 (b), the planar geometry of four microphones allows to localize a source in 3-D, with both azimuth and elevation [1]. More significantly, multiple coexisting sources $s^{\ell}(t)$ can be jointly separated and localized

using essentially the same principle [3, 4]:

$$\begin{aligned}
\xi_{00}(t) &= \sum_{\ell} s^{\ell}(t) + \nu_{00}(t) \\
\xi_{10}(t) &= \sum_{\ell} \tau_1^{\ell} \dot{s}^{\ell}(t) + \nu_{10}(t) \\
\xi_{01}(t) &= \sum_{\ell} \tau_2^{\ell} \dot{s}^{\ell}(t) + \nu_{01}(t)
\end{aligned} \tag{3}$$

where $\nu_{00}$, $\nu_{10}$ and $\nu_{01}$ represent common mode and spatial derivative components of additive noise in the sensor observations. Taking the time derivative of $\xi_{00}$, we thus obtain from the sensors a linear instantaneous mixture of the time-differentiated source signals,

$$\begin{bmatrix} \dot{\xi}_{00} \\ \dot{\xi}_{10} \\ \dot{\xi}_{01} \end{bmatrix} \approx \begin{bmatrix} 1 & \cdots & 1 \\ \tau_1^1 & \cdots & \tau_1^L \\ \tau_2^1 & \cdots & \tau_2^L \end{bmatrix} \begin{bmatrix} \dot{s}^1 \\ \vdots \\ \dot{s}^L \end{bmatrix} + \begin{bmatrix} \dot{\nu}_{00} \\ \nu_{10} \\ \nu_{01} \end{bmatrix}, \tag{4}$$

an equation in the standard form $\mathbf{x} = \mathbf{A}\mathbf{s} + \mathbf{n}$, where $\mathbf{x}$ is given and the mixing matrix $\mathbf{A}$ and sources $\mathbf{s}$ are unknown. Ignoring the noise term $\mathbf{n}$, the problem setting is standard in Independent Component Analysis (ICA), and three independent sources can be identified from the three gradient observations.

Various formulations of ICA exist to arrive at estimates of the unknown $\mathbf{s}$ and $\mathbf{A}$ from observations $\mathbf{x}$. ICA algorithms typically specify some sort of statistical independence assumption on the sources $\mathbf{s}$ either in distribution over amplitude [7] or over time [8]. Most forms specify ICA to be *static*, in assuming that the observations contain static (instantaneous) linear mixtures of the sources. Note that this definition of *static* ICA includes methods for blind source separation that make use of temporal structure in the dynamics within the sources themselves [8], as long as the observed mixture of the sources is static. In contrast, 'convolutive' ICA techniques explicitly assume convolutive or delayed mixtures in the source observations. Convolutive ICA techniques (*e.g.,* [10]) are usually much more involved and require a large number of parameters and long adaptation time horizons for proper convergence.

The instantaneous static formulation of gradient flow (4) is convenient,[2] and avoids the need for non-static (convolutive) ICA to separate delayed mixtures of traveling wave sources (in free space) $x_{pq}(t) = \sum_{\ell} s^{\ell}(t + p\tau_1 + q\tau_2)$. Reverberation in multipath wave propagation contributes delayed mixture components in the observations which limit the effectiveness of a static ICA formulation. As shown in the experiments below, static ICA still produces reasonable results (12 dB of perceptually clear separation) in typical enclosed acoustic environments (conference room).

## 3   Micropower VLSI Implementation

Various analog VLSI implementations of ICA exist in the literature, *e.g.,* [11, 12], and digital implementations using DSP are common practice in the field. By adopting a mixed-signal architecture in the implementation, we combine advantages of both approaches: an analog datapath directly interfaces with inputs and outputs without the need for data conversion; and digital adaptation offers the flexibility of reconfigurable ICA learning rules.

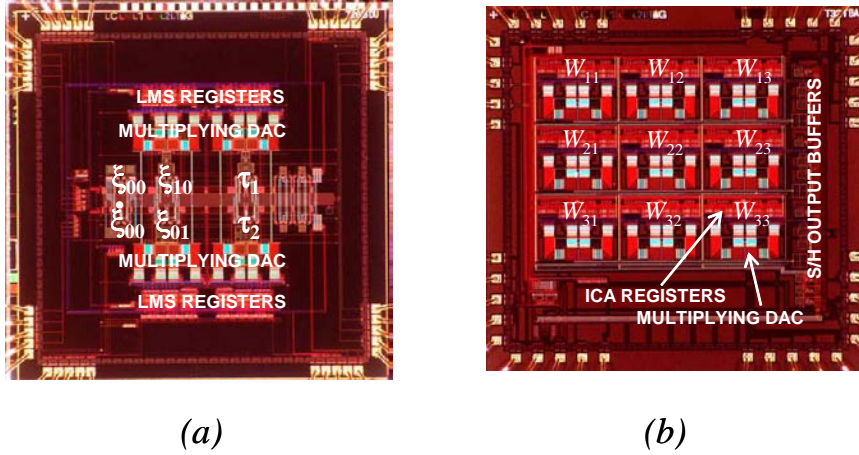

*(a)*                  *(b)*

Figure 2: *(a)* Gradient flow processor. *(b)* Reconfigurable ICA processor. Dimensions of both processors are 3mm $\times$ 3mm in 0.5 $\mu$m CMOS technology.

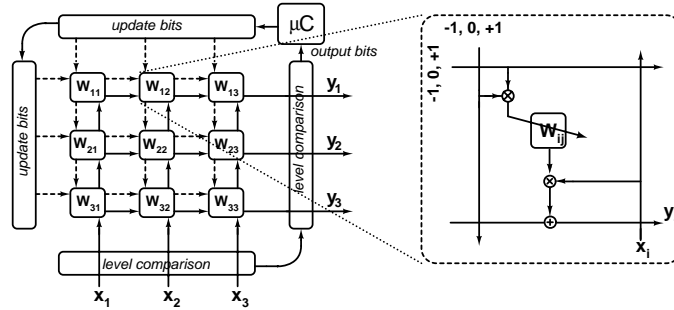

Figure 3: Reconfigurable mixed-signal ICA architecture implementing general outer-product forms of ICA update rules.

## 3.1 Gradient Flow Processor

The mixed-signal VLSI processor implementing gradient flow is presented in [6]. A micro-graph of the chip is shown in Figure 2 (a). Precise analog gradients $\dot{\xi}_{00}$, $\xi_{10}$ and $\xi_{01}$ are acquired from the microphone signals by correlated double sampling (CDS) in fully differential switched-capacitor circuits. Least-mean-squares (LMS) cancellation of common-mode leakage in the gradient signals further increases differential sensitivity. The adaptation is performed in the digital domain using counting registers, and couples to the switched-capacitor circuits using capacitive multiplying DAC arrays. An additional stage of LMS adaptation produces digital estimates of direction cosines $\tau_1$ and $\tau_2$ for a single source. In the present setup this stage is bypassed, and the common-mode corrected gradient signals are presented as inputs to the ICA chip for localization and separation of up to three independent sources.

## 3.2 Reconfigurable ICA Processor

A general mixed-signal parallel architecture, that can be configured for implementation of various ICA update rules in conjunction with gradient flow, is shown in Figure 3 [9]. Here

we briefly illustrate the architecture with a simple configuration designed to separate two sources, and present CMOS circuits that implement the architecture. The micrograph of the reconfigurable ICA chip is shown in Figure 2 (a).

### 3.2.1 ICA update rule

Efficient implementation in parallel architecture requires a simple form of the update rule, that avoids excessive matrix multiplications and inversions. A variety of ICA update algorithms can be cast in a common, unifying framework of outer-product rules [9].

To obtain estimates $\mathbf{y} = \hat{\mathbf{s}}$ of the sources $\mathbf{s}$, a linear transformation with matrix $\mathbf{W}$ is applied to the gradient signals $\mathbf{x}$, $\mathbf{y} = \mathbf{W}\mathbf{x}$. Diagonal terms are fixed $w_{ii} \equiv 1$, and off-diagonal terms adapt according to

$$\Delta w_{ij} = -\mu \, f(y_i)g(y_j), \qquad i \neq j \tag{5}$$

The implemented update rule can be seen as the gradient of *InfoMax* [7] multiplied by $\mathbf{W}^T$, rather than the natural gradient multiplication factor $\mathbf{W}^T\mathbf{W}$. To obtain the full natural gradient in outer-product form, it is necessary to include a back-propagation path in the network architecture, and thus additional silicon resources, to implement the vector contribution $\mathbf{y}^T$. Other equivalences with standard ICA algorithms are outlined in [9].

### 3.2.2 Architecture

Level comparison provides implementation of discrete approximations of any scalar function $f(y)$ and $g(y)$ appearing in different learning rules. Since speech signals are approximately Laplacian distributed, the nonlinear scalar function $f(y)$ is approximated by $\text{sign}(y)$ and implemented using single bit quantization. Conversely, a linear function $g(y) \equiv y$ in the learning rule is approximated by a 3-level staircase function $(-1, 0, +1)$ using 2-bit quantization. The quantization of the $f$ and $g$ terms in the update rule (5) simplifies the implementation to that of discrete counting operations.

The functional block diagram of a $3 \times 3$ outer-product incremental ICA architecture, supporting a quantized form of the general update rule (5), is shown in Figure 3 [9]. Un-mixing coefficients are stored digitally in each cell of the architecture. The update is performed locally by once or repeatedly incrementing, decrementing or holding the current value of counter based on the learning rule served by the micro-controller. The 8 most significant bits of the 14-bit counter holding and updating the coefficients are presented to a multiplying D/A capacitor array [6] to linearly unmix the separated signal. The remaining 6 bits in the coefficient registers provide flexibility in programming the update rate to tailor convergence.

### 3.2.3 Circuit implementation

As in the implementation of the gradient flow processor [6], the mixed-signal ICA architecture is implemented using fully differential switched-capacitor sampled-data circuits. Correlated double sampling performs common mode offset rejection and 1/f noise reduction. An external micro-controller provides flexibility in the implementation of different learning rules. The ICA architecture is integrated on a single $3mm \times 3mm$ chip fabricated in 0.5 $\mu$m 3M2P CMOS technology.

The block diagram of ICA prototype in Figure 3 indicates its main functionality is a vector(3x1)-matrix(3x3) multiplication with adaptive matrix elements.

Each cell in the implemented architecture contains a 14-bit counter, decoder and D/A capacitor arrays. Adaptation is performed in outer-product fashion by incrementing, decrementing or holding the current value of the counters. The most significant 8 bits of the

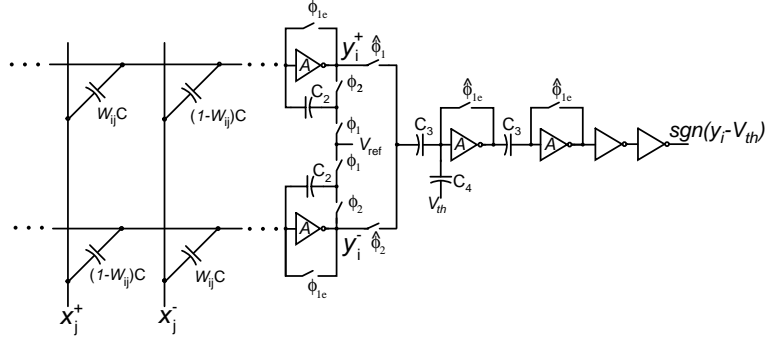

Figure 4: Correlated double sampling (CDS) switched-capacitor fully differential circuits implementing linearly weighted summing in the mixed-signal ICA architecture.

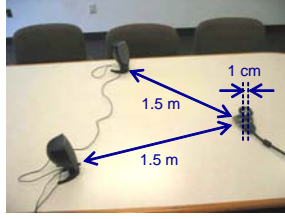

Figure 5: Experimental setup for separation of two acoustic sources in a conference room enviroment.

counter are presented to the multiplying D/A capacitor arrays to construct the source estimation. Figure 4 shows the circuits one output component in the architecture, linearly summing the input contributions. The implementation of the multiplying capacitor arrays are identical to those discussed in [6]. Each output signal $y_i$ is is computed by accumulating outputs from the all the cells in the $i^{th}$row. The accumulation is performed on $C_2$ by switch-cap amplifier yielding the estimated signals during $\Phi_2$ phase. While the estimation signals are valid, $y_i^+$ is sampled at $\hat{\Phi}_1$ by the comparator circuit. The sign of the comparison of $y_i$ with variable level threshold $V_{th}$ is computed in the evaluate phase, through capacitive coupling into the amplifier input node.

## 4   Experimental Results

To demonstrate source separation and localization in a real environment, the mixed-signal VLSI ASICs were interfaced with four omnidirectional miniature microphones (Knowles FG-3629), arranged in a circular array with radius 0.5 cm. At the front-end, the microphone signals were passed through second-order bandpass filters with low-frequency cutoff at 130 Hz and high-frequency cutoff at 4.3 kHz. The signals were also amplified by a factor of 20.

The experimental setup is shown in Figure 5. The speech signals were presented through loudspeakers positioned at 1.5 m distance from the array. The system sampling frequency of both chips was set to 16 kHz. A male and female speakers from TIMIT database were chosen as sound sources. To provide the ground truth data and full characterization of the systems, speech segments were presented individually through either loudspeaker at different time instances. The data was recorded for both speakers, archived, and presented to the

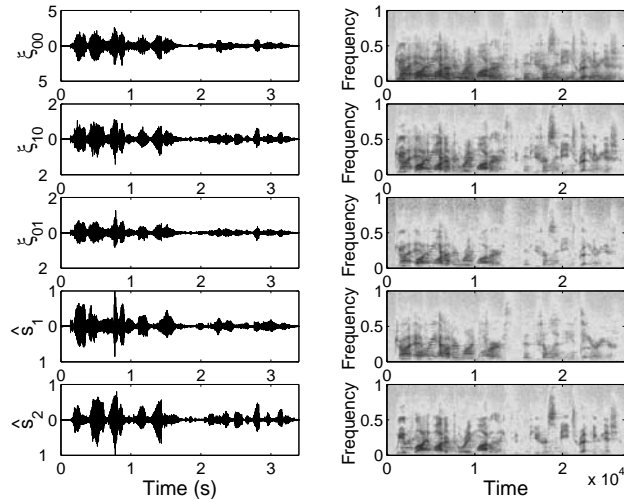

Figure 6: Time waveforms and spectrograms of the presented sources $s_1$ and $s_2$, observed common-mode and gradient signals $\xi_{00}$, $\xi_{10}$ and $\xi_{01}$ by the gradient flow chip, and recovered sources $\hat{s}_1$ and $\hat{s}_2$ by the ICA chip.

Table 1: Localization Performance

|  | Male speaker | Female speaker |
|---|---|---|
| Single-source LMS localization | -31.11 | 40.95 |
| Dual-source ICA localization | -30.35 | 43.55 |

gradient flow chip. Localization results obtained by gradient flow chip through LMS adaptation are reported in Table 1. The two recorded datasets were then added, and presented to the gradient flow ASIC. The gradient signals obtained from the chip were then presented to the ICA processor, configured to implement the outerproduct update algorithm in (5). The observed convergence time was around 2 seconds. From the recorded 14-bit digital weights, the angles of incidence of the sources relative to the array were derived. These estimated angles are reported in Table 1. As seen, the angles obtained through LMS bearing estimation under individual source presentation are very close to the angles produced by ICA under joint presentation of both sources. The original sources and the recorded source signal estimates, along with recorded common-mode signal and first-order spatial gradients, are shown in Figure 6.

## 5  Conclusions

We presented a mixed-signal VLSI system that operates on spatial and temporal differences (gradients) of the acoustic field at very small aperture to separate and localize mixtures of traveling wave sources. The real-time performance of the system was characterized through experiments with speech sources presented through speakers in a conference room setting. Although application of static ICA is limited by reverberation, the perceptual quality of the separated outputs owes to the elimination of the direct path in the residuals. Miniature size of the microphone array enclosure (1 cm diameter) and micropower consumption of the VLSI hardware (250 $\mu$W) are key advantages of the approach, with applications to hearing

aids, conferencing, multimedia, and surveillance.

**Acknowledgments**

This work was supported by grants of the Catalyst Foundation (New York), the National Science Foundation, and the Defense Intelligence Agency.

## Footnotes

[1] An alternative using two microphones, exploiting shape of the pinna, is presented in [5]

[2]The time-derivative in the source signals (4) is immaterial, and can be removed by time-integrating the separated signals obtained by applying ICA directly to the gradient flow signals.

# References

[1] D. Robert, R.N. Miles, and R.R. Hoy, "Tympanal Hearing in the Sarcophagid Parasitoid Fly Emblemasoma sp.: the Biomechanics of Directional Hearing," *J. Experimental Biology*, vol. 202, pp 1865-1876, 1999.

[2] R. Reeve and B. Webb, "New neural circuits for robot phonotaxis", *Philosophical Transactions of the Royal Society A*, vol. **361**, pp. 2245-2266, 2002.

[3] G. Cauwenberghs, M. Stanacevic, and G. Zweig, "Blind Broadband Source Localization and Separation in Miniature Sensor Arrays," *Proc. IEEE Int. Symp. Circuits and Systems (ISCAS'2001),* Sydney, Australia, May 6-9, 2001.

[4] J. Barrère and G. Chabriel, "A Compact Sensor Array for Blind Separation of Sources", *IEEE Transactions Circuits and Systems, Part I*, vol. **49** (5), pp. 565-574, 2002.

[5] J.G. Harris, C.-J. Pu, J.C. Principe, "A Neuromorphic Monaural Sound Localizer," *Proc. Neural Inf. Proc. Sys. (NIPS*1998)*, Cambridge MA: MIT Press, vol. 10, pp. 692-698, 1999.

[6] G. Cauwenberghs and M. Stanacevic, "Micropower Mixed-Signal Acoustic Localizer," *Proc. IEEE Eur. Solid State Circuits Conf. (ESSCIRC'2003)*, Estoril Portugal, Sept. 16-18, 2003.

[7] A.J. Bell and T.J. Sejnowski, "An Information Maximization Approach to Blind Separation and Blind Deconvolution," *Neural Comp*, vol. **7** (6), pp 1129-1159, Nov 1995.

[8] L. Molgedey and G. Schuster, "Separation of a mixture of independent signals using time delayed correlations," *Physical Review Letters*, vol. 72, no. 23, pp. 3634–3637, 1994.

[9] A. Celik, M. Stanacevic and G. Cauwenberghs, "Mixed-Signal Real-Time Adaptive Blind Source Separation," *Proc. IEEE Int. Symp. Circuits and Systems (ISCAS'2004),* Vancouver Canada, May 23-26, 2004.

[10] R. Lambert and A. Bell, "Blind separation of multiple speakers in a multipath environment," *Proc. ICASSP'97*, Münich, 1997.

[11] Cohen, M.H., Andreou, A.G. "Analog CMOS Integration and Experimentation with an Autoadaptive Independent Component Analyzer," *IEEE Trans. Circuits and Systems II,* vol 42 (2), pp 65-77, Feb. 1995.

[12] Gharbi, A.B.A., Salam, F.M.A. "Implementation and Test Results of a Chip for the Separation of Mixed Signals," *Proc. Int. Symp. Circuits and Systems (ISCAS'95)*, May 1995.

[13] M. Cohen and G. Cauwenberghs, "Blind Separation of Linear Convolutive Mixtures through Parallel Stochastic Optimization," *Proc. IEEE Int. Symp. Circuits and Systems (ISCAS'98),* Monterey CA, vol. 3, pp. 17-20, 1998.
